# How the Poverty of the Stimulus Solves the Poverty of the Stimulus

**Willem Zuidema**
Language Evolution and Computation Research Unit
and Institute for Cell, Animal and Population Biology
University of Edinburgh
40 George Square, Edinburgh EH8 9LL, United Kingdom
jelle@ling.ed.ac.uk

## Abstract

Language acquisition is a special kind of learning problem because the outcome of learning of one generation is the input for the next. That makes it possible for languages to *adapt* to the particularities of the learner. In this paper, I show that this type of language change has important consequences for models of the evolution and acquisition of syntax.

## 1    The Language Acquisition Problem

For both artificial systems and non-human animals, learning the syntax of natural languages is a notoriously hard problem. All healthy human infants, in contrast, learn any of the approximately 6000 human languages rapidly, accurately and spontaneously. Any explanation of how they accomplish this difficult task must specify the (innate) *inductive bias* that human infants bring to bear, and the input data that is available to them. Traditionally, the inductive bias is termed – somewhat unfortunately – "Universal Grammar", and the input data "primary linguistic data".

Over the last 30 years or so, a view on the acquisition of the syntax of natural language has become popular that has put much emphasis on the innate machinery. In this view, that one can call the "Principles and Parameters" model, the Universal Grammar specifies most aspects of syntax in great detail [e.g. 1]. The role of experience is reduced to setting a limited number (30 or so) of parameters. The main argument for this view is the *argument from the poverty of the stimulus* [2]. This argument states that children have insufficient evidence in the primary linguistic data to induce the grammar of their native language.

Mark Gold [3] provides the most well-known formal basis to this argument. Gold introduced the criterion "identification in the limit" for evaluating the success of a learning algorithm: with an infinite number of training samples all hypotheses of the algorithm should be identical, and equivalent to the target. Gold showed that the class of context-free grammars is not learnable in this sense by any algorithm from positive samples alone (and neither are other *super-finite* classes). This proof is based on the fact that no matter how many samples from an infinite language a

learning algorithm has seen, the algorithm can not decide *with certainty* that the samples are drawn from the infinite language or from a finite language that contains all samples. Because natural languages are thought to be at least as complex as context-free grammars, and negative feedback is assumed to be absent in the primary linguistic data, Gold's analysis, and subsequent work in learnability theory [1], is usually interpreted as strong support for the argument from the poverty of the stimulus, and, in the extreme, for the view that grammar induction is fundamentally impossible (a claim that Gold would not subscribe to).

Critics of this "nativist" approach [e.g. 4, 5] have argued for different assumptions on the appropriate grammar formalism (e.g. stochastic context-free grammars), the available primary data (e.g. semantic information) or the appropriate learnability criterion. In this paper I will take a different approach. I will present a model that induces *context-free grammars* without a-priori restrictions on the search space, semantic information or negative evidence. Gold's negative results thus apply. Nevertheless, acquisition of grammar is successful in my model, because another process is taken into account as well: the cultural evolution of language.

## 2 The Language Evolution Problem

Whereas in language acquisition research the central question is how a child acquires an *existing* language, in language evolution research the central question is how this language and its properties have emerged in the first place. Within the nativist paradigm, some have suggested that the answer to this question is that Universal Grammar is the product of evolution under selection pressures for communication [e.g. 6]. Recently, several formal models have been presented to evaluate this view. For this paper, the most relevant of those is the model of Nowak et al. [7].

In that model it is assumed that there is a finite number of grammars, that newcomers (infants) learn their grammar from the population, that more successful grammars have a higher probability of being learned and that mistakes are made in learning. The system can thus be described in terms of the changes in the relative frequencies $x_i$ of each grammar type $i$ in the population. The first result that Nowak et al. obtain is a "coherence threshold". This threshold is the necessary condition for grammatical coherence in a population, i.e. for a majority of individuals to use the same grammar. They show that this coherence depends on the chances that a child has to correctly acquire its parents' grammar. This probability is described with the parameter $q$. Nowak et al. show analytically that there is a minimum value for $q$ to keep coherence in the population. If $q$ is lower than this value, all possible grammar types are equally frequent in the population and the communicative success in minimal. If $q$ is higher than this value, one grammar type is dominant; the communicative success is much higher than before and reaches 100% if $q = 1$.

The second result relates this required fidelity (called $q_1$) to a lower bound ($b_c$) on the number of sample sentences that a child needs. Nowak et al. make the crucial assumption that all languages are equally expressive and equally different from each other. With that assumption they can show that $b_c$ is proportional to the total number of possible grammars $N$. Of course, the actual number of sample sentences $b$ is finite; Nowak et al. conclude that only if $N$ is relatively small can a stable grammar emerge in a population. I.e. the population dynamics require a restrictive Universal Grammar.

The models of Gold and Nowak et al. have in common that they implicitly assume that every possible grammar is equally likely to become the target grammar for learning. If even the best possible learning algorithm cannot learn such a grammar,

the set of allowed grammars must be restricted. There is, however, reason to believe that this assumption is not the most useful for language learning. Language learning is a very particular type of learning problem, because the outcome of the learning process at one generation is the input for the next. The samples from which a child learns with its learning procedure, are therefore *biased* by the learning of previous generations that used the same procedure[8].

In [9] and other papers, Kirby, Hurford and students have developed a framework to study the consequences of that fact. In this framework, called the "Iterated Learning Model" (ILM), a population of individuals is modeled that can each produce and interpret sentences, and have a language acquisition procedure to learn grammar from each other. In the ILM one individual (the parent) presents a relatively small number of examples of form–meaning pairs to the next individual (the child). The child then uses these examples to induce his own grammar. In the next iteration the child becomes the parent, and a new individual becomes the child. This process is repeated many times. Interestingly, Kirby and Hurford have found that in these iterated transmission steps the language becomes easier and easier to learn, because the language adapts to the learning algorithm by becoming more and more structured. The structure of language in these models thus emerges from the iteration of learning. The role of biological evolution, in this view, is to shape the learning algorithms, such that the complex results of the iterated learning is biologically adaptive [10]. In this paper I will show that if one adopts this view on the interactions between learning, cultural evolution and biological evolution, the models such as those of Gold [3] and Nowak et al. [7] can no longer be taken as evidence for an extensive, innate pre-specification of human language.

## 3   A Simple Model of Grammar Induction

To study the interactions between language adaptation and language acquisition, I have first designed a grammar induction algorithm that is simple, but can nevertheless deal with some non-trivial induction problems. The model uses context-free grammars to represent linguistic abilities. In particular, the representation is limited to grammars $G$ where all rules are of one of the following forms: (1) $A \mapsto t$, (2) $A \mapsto BC$, (3) $A \mapsto Bt$. The nonterminals $A, B, C$ are elements of the non-terminal alphabet $V_{nt}$, which includes the start symbol $S$. $t$ is a string of terminal symbols from the terminal alphabet $V_t$[1]. For determining the language $L$ of a certain grammar $G$ I use simple depth-first exhaustive search of the derivation tree. For computational reasons, the depth of the search is limited to a certain depth $d$, and the string length is limited to length $l$. The set of sentences ($L' \subseteq L$) used in training and in communication is therefore finite (and strictly speaking not context-free, but regular); in production, strings are drawn from a uniform distribution over $L'$.

The grammar induction algorithm learns from a set of sample strings (sentences) that are provided by a teacher. The design of the learning algorithm is originally inspired by [11] and is similar to the algorithm in [12]. The algorithm fits within a tradition of algorithms that search for compact descriptions of the input data [e.g. 13, 14, 15]. It consists of three operations:

**Incorporation:** *extend the language, such that it includes the encountered string;* if string $s$ is not already part of the language, add a rule $S \mapsto s$ to the grammar.

**Compression:** *substitute frequent and long substrings with a nonterminal, such that the grammar becomes smaller and the language remains unchanged;* for every valid substring $z$ of the right-hand sides of all rules, calculate the compression effect $v(z)$ of substituting $z$ with a nonterminal $A$; replace all valid occurrences of the substring $z' = argmax_z v(z)$ with $A$ if $v(z') > 0$, and add a rule $A \mapsto z'$ to the grammar. "Valid substrings" are those substrings which can be replaced while keeping all rules of the forms 1–3 described above. The compression effect is measured as the difference between the number of symbols in the grammar before and after the substitution. The compression step is repeated until the grammar does not change anymore.

**Generalization:** *equate two nonterminals, such that the grammar becomes smaller and the language larger;* for every combination of two nonterminals $A$ and $B$ ($B \neq S$), calculate the compression effect $v$ of equating $A$ and $B$. Equate the combination $(A', B') = argmax_{A,B} v(A, B)$ if $v(A', B') > 0$; i.e. replace all occurrences of $B$ with $A$. The compression effect is measured as the difference between the number of symbols before and after replacing and deleting redundant rules. The generalization step is repeated until the grammar does not change anymore.

## 4  Learnable and Unlearnable Classes

The algorithm described above is implemented in $C^{++}$ and tested on a variety of target grammars[2]. I will not present a detailed analysis of the learning behavior here, but limit myself to a simple example that shows that the algorithm can learn some (recursive) grammars, while it can not learn others. The induction algorithm receives three sentences (abcd, abcabcd, abcabcabcd). The incorporation, compression (repeated twice) and generalization steps yield subsequently the following grammars:

| (a) Incorporation | | | (b) Compression | | | (c) Generalization | | |
|---|---|---|---|---|---|---|---|---|
| S | $\mapsto$ | abcd | S | $\mapsto$ | Yd | S | $\mapsto$ | Xd |
| S | $\mapsto$ | abcabcd | S | $\mapsto$ | Xd | S | $\mapsto$ | Xabcd |
| S | $\mapsto$ | abcabcabcd | S | $\mapsto$ | Xabcd | X | $\mapsto$ | XX |
| | | | X | $\mapsto$ | YY | X | $\mapsto$ | abc |
| | | | Y | $\mapsto$ | abc | | | |

In (b) the substrings "abcabc" and "abc" are subsequently replaced by the nonterminals X and Y. In (c) the non-terminals X and Y are equated, which leads to the deletion of the second rule in (b). One can check that the total size of the grammar reduces from 24, to 19 and further down to 16 characters.

From this example it is also clear that learning is not always successful. Any of the three grammars above ((a) and (b) are equivalent) could have generated the training data, but with these three input strings the algorithm always yields grammar (c). Consistent with Gold's general proof [3], many target grammars will never be learned correctly, no matter how many input strings are generated. In practice, each finite set of randomly generated strings from some target grammar, might yield a different result. Thus, for some number of input strings $T$, some set of target grammars are always acquired, some are never acquired, and some are some of the time acquired. If we can enumerate all possible grammars, we can describe this with a matrix $Q$, where each entry $Q_{ij}$ describes the probability that the algorithm learning from sample strings from a target grammar $i$, will end up with grammar

of type $j$. $Q_{ii}$ is the probability that the algorithm finds the target grammar. To make learning successful, the target grammars that are presented to the algorithm have to be biased. The following section will show that for this we need nothing more than to assume that the output of one learner is the input for the next.

# 5  Iterated Learning: the Emergence of Learnability

To study the effects of iterated learning, we extend the model with a population structure. In the new version of the model individuals (agents, that each represent a generation) are placed in a *chain*. The first agent induces its grammar from a number $E$ of randomly generated strings. Every subsequent agent (the child) learns its grammar from $T$ sample sentences that are generated by the previous one (the parent). To avoid insufficient expressiveness, we also extend the generalization step with a check if the number $E_G$ of different strings the grammar $G$ can recognize is larger than or equal to E. If not, $E - E_G$ random new strings are generated and incorporated in the grammar. Using the matrix $Q$ from the previous section, we can formalize this *iterated learning model* with the following general equation, where $x_i$ is the probability that grammar $i$ is the grammar of the current generation:

$$\Delta x_i = \sum_{j=0}^{N} x_j Q_{ji} \qquad (1)$$

In simulations such as the one of figure 1 communicative success between child and parent – a measure for the learnability of a grammar – rises steadily from a low value (here 0.65) to a high value (here 1.0). In the initial stage the grammar shows no structure, and consequently almost every string that the grammar produces is idiosyncratic. A child in this stage typically hears strings like "ada", "ddac", "adba", "bcbd", or "cdca" from its parent. It can not discover many regularities in these strings. The child therefore can not do much better than simply reproduce the strings it heard (i.e. $T$ random draws from at least $E$ different strings), and generate random new strings, if necessary to make sure its language obeys the minimum number ($E$) of strings. However, in these randomly generated strings, sometimes regularities appear. I.e., a parent may use the randomly generated strings "dcac", "bcac", "caac" and "daac". When this happens the child tends to analyze these strings as different combinations with the building block "ac". Thus, typically, the learning algorithm generates a grammar with the rules $S \mapsto dcX$, $S \mapsto bcX$, $S \mapsto caX$, $S \mapsto daX$, and $X \mapsto ac$. When this happens to another set of strings as well, say with a new rule $Y \mapsto b$, the generalization procedure can decide to equate the non-terminals $X$ and $Y$. The resulting grammar can then generalize from the observed strings, to the unobserved strings "dcb", "bcb", "cab" and "dab". The child still needs to generate random new strings to reach the minimum $E$, but fewer than in the case considered above.

The interesting aspect of this becomes clear when we consider the next step in the simulation, when the child becomes itself the parent of a new child. This child is now presented with a language with more regularities than before, and has a fair chance of *correctly* generalizing to unseen examples. If, for instance, it only sees the strings "dcac", "bcac", "caac", "bcb", "cab" and "dab", it can, through the same procedure as above, infer that "daac" and "dcb" are also part of the target language. This means that (i) the child shares more strings with its parent than just the ones it observes and consequently shows a higher between generation communicative success, and (ii) regularities that appear in the language by chance, have a fair chance to remain in the language. In the process of iterated learning, languages can thus become more structured and better learnable.

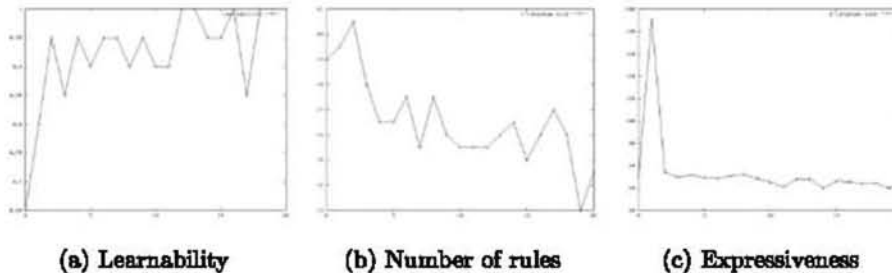

| (a) Learnability | (b) Number of rules | (c) Expressiveness |

Figure 1: Iterated Learning: although initially the target language is unstructured and difficult to learn, over the course of 20 generations (a) the learnability (the fraction of successful communications with the parent) steadily increases, (b) the number of rules steadily decreases (combinatorial and recursive stategies are used), and (c) after a initial phase of overgeneralization, the expressiveness remains close to its minimally required level. Parameters: $V_t = \{a, b, c, d\}$, $V_{nt} = \{S, X, Y, Z, A, B, C\}$, T=30, E=20, $l_0$=3. Shown are the average values of 2 simulations.

Similar results with different formalisms were already reported before [e.g. 11, 16], but here I have used context-free grammars and the results are therefore directly relevant for the interpretation of Gold's proof [3]. Whereas in the usual interpretation of that proof [e.g. 1] it is assumed that we need innate constraints on the *search space* in addition to a smart *learning procedure*, here I show that even a simple learning procedure can lead to successful acquisition, because restrictions on the search space automatically emerge in the iteration of learning. If one considers learnability a *binary* feature – as is common in generative linguistics – this is a rather trivial phenomenon: languages that are not learnable will not occur in the next generation. However, if there are gradations in learnability, the cultural evolution of language can be an intricate process where languages get shaped over many generations.

## 6  Language Adaptation and the Coherence Threshold

When we study this effect in a version of the model where *selection* does play a role, it is also relevant for the analysis in [7]. The model is therefore extended such that at every generation there is a population of agents, agents of one generation communicate with each other and the expected number of offspring of an agent (the *fitness*) is determined by the number of successful interactions it had. Children still acquire their grammar from sample strings produced by their parent. Adapting equation 1, this system can now be described with the following equation, where $x_i$ is now the relative fraction of grammar $i$ in the population (assuming an infinite population size):

$$\Delta x_i = \sum_{j=0}^{N} x_j f_j Q_{ji} - \phi x_i \tag{2}$$

Here, $f_i$ is the *relative fitness* (quality) of grammars of type $i$ and equals $f_i = \sum_j x_j F_{ij}$, where $F_{ij}$ is the expected communicative success from an interaction between an individual of type $i$ and an individual of type $j$. The relative fitness $f$ of a grammar thus depends on the frequencies of all grammar types, hence it is *frequency dependent*. $\phi$ is the average fitness in the population and equals $\phi = \sum_i x_i f_i$. This

term is needed to keep the sum of all fractions at 1. This equation is essentially the model of Nowak et al. [7]. Recall that the main result of that paper is a "coherence threshold": a minimum value for the learning accuracy $q$ to keep coherence in the population. In previous work [unpublished] I have reproduced this result and shown that it is robust against variations in the $Q$-matrix, as long as the value of $q$ (i.e. the diagonal values) remains equal for all grammars.

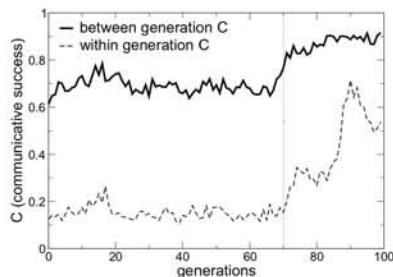

Figure 2: Results from a run under fitness proportional selection. This figure shows that there are regions of grammar space where the dynamics are apparently under the "coherence threshold" [7], while there are other regions where the dynamics are above this threshold. The parameters, including the number of sample sentences $T$, are still the same, but the language has adapted itself to the **bias** of the learning algorithm. Parameters are: $V_t = \{0, 1, 2, 3\}$, $V_{nt} = \{S, a, b, c, d, e, f\}$, P=20, T=100, E=100, $l_0$=12. Shown are the average values of 20 agents.

Figure 2, however, shows results from a simulation with the grammar induction algorithm described above, where this condition is violated. Whereas in the simulations of figure 1 the target languages have been relatively easy (the initial string length is short, i.e. 6), here the learning problem is very difficult (initial string length is long, i.e. 12). For a long period the learning is therefore not very successful, but around generation 70 the success suddenly rises. With always the same $T$ (number of sample sentences), and with always the same grammar space, there are regions where the dynamics are apparently under the "coherence threshold", while there are other regions where the dynamics are above this threshold. The language has adapted to the learning algorithm, and, consequently, the coherence in the population does not satisfy the prediction of Nowak et al.

## 7 Conclusions

I believe that these results have some important consequences for our thinking about language acquisition. In particular, they offer a different perspective on the argument from the poverty of the stimulus, and thus on one of the most central "problems" of language acquisition research: *the logical problem of language acquisition*. My results indicate that in *iterated learning* it is not necessary to put the (whole) explanatory burden on the representation bias. Although the details of the grammatical formalism (context-free grammars) and the population structure are deliberately close to [3] and [7] respectively, I do observe successful acquisition of grammars from a class that is unlearnable by Gold's criterion. Further, I observe grammatical coherence even though many more grammars are allowed in principle than Nowak et al. calculate as an upper bound. The reason for these surprising results is that language acquisition is a very particular type of learning problem: it is a problem where the target of the learning process is itself the outcome of a learning process. That opens up the possibility of language itself to adapt to the

language acquisition procedure of children. In such iterated learning situations [11], learners are only presented with targets that other learners have been able to learn.

Isn't this the traditional Universal Grammar in disguise? Learnability is – consistent with the undisputed proof of [3] – still achieved by constraining the set of targets. However, unlike in usual *interpretations* of this proof, these constraints are not strict (some grammars are better learnable than others, allowing for an infinite "Grammar Universe"), and they are not a-priori: they are the outcome of iterated learning. The poverty of the stimulus is now no longer a problem; instead, the ancestors' poverty is the solution for the child's.

**Acknowledgments**   This work was performed while I was at the AI Laboratory of the Vrije Universiteit Brussel. It builds on previous work that was done in close collaboration with Paulien Hogeweg of Utrecht University. I thank her and Simon Kirby, John Batali, Aukje Zuidema and my colleagues at the AI Lab and the LEC for valuable hints, questions and remarks. Funding from the Concerted Research Action fund of the Flemish Government and the VUB, from the Prins Bernhard Cultuurfonds and from a Marie Curie Fellowship of the European Commission are gratefully acknowledged.

## Footnotes

[1]Note that the restrictions on the rule-types above do not limit the scope of languages that can be represented (they are essentially equivalent to Chomsky Normal Form). They are, however, relevant for the language acquisition algorithm.

[2]The source code is available at http://www.ling.ed.ac.uk/~jelle

## References

[1] Stefano Bertolo, editor. *Language Acquisition and Learnability*. Cambridge University Press, 2001.

[2] Noam Chomsky. *Aspects of the theory of syntax*. MIT Press, Cambridge, MA, 1965.

[3] E. M. Gold. Language identification in the limit. *Information and Control (now Information and Computation)*, 10:447–474, 1967.

[4] Michael A. Arbib and Jane C. Hill. Language acquisition: Schemas replace universal grammar. In John A. Hawkins, editor, *Explaining Language Universals*. Basil Blackwell, New York, USA, 1988.

[5] J. Elman, E. Bates, et al. *Rethinking innateness*. MIT Press, 1996.

[6] Steven Pinker and Paul Bloom. Natural language and natural selection. *Behavioral and brain sciences*, 13:707–784, 1990.

[7] Martin A. Nowak, Natalia Komarova, and Partha Niyogi. Evolution of universal grammar. *Science*, 291:114–118, 2001.

[8] Terrence Deacon. *Symbolic species, the co-evolution of language and the human brain.* The Penguin Press, 1997.

[9] S. Kirby and J. Hurford. The emergence of linguistic structure: An overview of the iterated learning model. In Angelo Cangelosi and Domenico Parisi, editors, *Simulating the Evolution of Language*, chapter 6, pages 121–148. Springer Verlag, London, 2002.

[10] Kenny Smith. Natural selection and cultural selection in the evolution of communication. *Adaptive Behavior*, 2003. to appear.

[11] Simon Kirby. Syntax without natural selection: How compositionality emerges from vocabulary in a population of learners. In C. Knight et al., editors, *The Evolutionary Emergence of Language*. Cambridge University Press, 2000.

[12] J. Gerard Wolff. Language acquisition, data compression and generalization. *Language & Communication*, 2(1):57–89, 1982.

[13] A. Stolcke. *Bayesian Learning of Probabilistic Language Models*. PhD thesis, Dept. of Electrical Engineering and Computer Science, University of California at Berkeley, 1994.

[14] Menno van Zaanen and Pieter Adriaans. Comparing two unsupervised grammar induction systems: Alignment-based learning vs. EMILE. In Ben Kröse et al., editors, *Proceedings of BNAIC 2001*, 2001.

[15] Zach Solan, Eytan Ruppin, David Horn, and Shimon Edelman. Automatic acquisition and efficient representation of syntactic structures. *This volume*.

[16] Henry Brighton. Compositional syntax from cultural transmission. *Artificial Life*, 8(1), 2002.
